# Information Diffusion Kernels

**John Lafferty**
School of Computer Science
Carnegie Mellon University
Pittsburgh, PA 15213 USA
lafferty@cs.cmu.edu

**Guy Lebanon**
School of Computer Science
Carnegie Mellon University
Pittsburgh, PA 15213 USA
lebanon@cs.cmu.edu

## Abstract

A new family of kernels for statistical learning is introduced that exploits the geometric structure of statistical models. Based on the heat equation on the Riemannian manifold defined by the Fisher information metric, information diffusion kernels generalize the Gaussian kernel of Euclidean space, and provide a natural way of combining generative statistical modeling with non-parametric discriminative learning. As a special case, the kernels give a new approach to applying kernel-based learning algorithms to discrete data. Bounds on covering numbers for the new kernels are proved using spectral theory in differential geometry, and experimental results are presented for text classification.

## 1 Introduction

The use of kernels is of increasing importance in machine learning. When "kernelized," simple learning algorithms can become sophisticated tools for tackling nonlinear data analysis problems. Research in this area continues to progress rapidly, with most of the activity focused on the underlying learning algorithms rather than on the kernels themselves.

Kernel methods have largely been a tool for data represented as points in Euclidean space, with the collection of kernels employed limited to a few simple families such as polynomial or Gaussian RBF kernels. However, recent work by Kondor and Lafferty [7], motivated by the need for kernel methods that can be applied to discrete data such as graphs, has proposed the use of diffusion kernels based on the tools of spectral graph theory. One limitation of this approach is the difficulty of analyzing the associated learning algorithms in the discrete setting. For example, there is no obvious way to bound covering numbers and generalization error for this class of diffusion kernels, since the natural function spaces are over discrete sets.

In this paper, we propose a related construction of kernels based on the heat equation. The key idea in our approach is to begin with a statistical model of the data being analyzed, and to consider the heat equation on the Riemannian manifold defined by the Fisher information metric of the model. The result is a family of kernels that naturally generalizes the familiar Gaussian kernel for Euclidean space, and that includes new kernels for discrete data by beginning with statistical families such as the multinomial. Since the kernels are intimately based on the geometry of the Fisher information metric and the heat or diffusion equation on the associated Riemannian manifold, we refer to them as *information diffusion kernels*.

Unlike the diffusion kernels of [7], the kernels we investigate here are over continuous parameter spaces even in the case where the underlying data is discrete. As a consequence, some of the machinery that has been developed for analyzing the generalization performance of kernel machines can be applied in our setting. In particular, the spectral approach of Guo *et al.* [3] is applicable to information diffusion kernels, and in applying this approach it is possible to draw on the considerable body of research in differential geometry that studies the eigenvalues of the geometric Laplacian.

In the following section we review the relevant concepts that are required from information geometry and classical differential geometry, define the family of information diffusion kernels, and present two concrete examples, where the underlying statistical models are the multinomial and spherical normal families. Section 3 derives bounds on the covering numbers for support vector machines using the new kernels, adopting the approach of [3]. Section 4 describes experiments on text classification, and Section 5 discusses the results of the paper.

## 2  Information Geometry and Diffusion Kernels

Let $\mathcal{S} = \{p(\cdot \mid \theta), \; \theta \in \Theta \subset \mathbb{R}^d\}$ be a $d$-dimensional statistical model on a set $\mathcal{X}$. For each $x \in \mathcal{X}$ assume the mapping $\theta \mapsto p(x \mid \theta)$ is $C^\infty$ at each point in the interior of $\Theta$. Let $\partial_i = \frac{\partial}{\partial \theta_i}$ and $\ell_\theta(x) = \log p(x \mid \theta)$. The Fisher information matrix $[g_{ij}(\theta)]$ of $\mathcal{S}$ at $\theta \in \Theta$ is given by

$$g_{ij}(\theta) = E_\theta[\partial_i \ell_\theta \, \partial_j \ell_\theta] = \int_{\mathcal{X}} \partial_i \log p(x \mid \theta) \, \partial_j \log p(x \mid \theta) \, p(x \mid \theta) \, dx \tag{1}$$

or equivalently as

$$g_{ij}(\theta) = 4 \int_{\mathcal{X}} \partial_i \sqrt{p(x \mid \theta)} \, \partial_j \sqrt{p(x \mid \theta)} \, dx \,. \tag{2}$$

In coordinates $\theta_i$, $g_{ij}(\theta)$ defines a Riemannian metric on $\Theta$, giving $\mathcal{S}$ the structure of a $d$-dimensional Riemannian manifold. One of the motivating properties of the Fisher information metric is that, unlike the Euclidean distance, it is invariant under reparameterization. For detailed treatments of information geometry we refer to [1, 6].

For many statistical models there is a natural way to associate to each data point $x$ a parameter vector $\theta(x)$ in the statistical model. For example, in the case of text, under the multinomial model a document is naturally associated with the relative frequencies of the word counts. This amounts to the mapping which sends a document $x$ to its maximum likelihood model $\hat{\theta}(x)$. Given such a mapping, we propose to apply a kernel on parameter space, $K_t(x, x') \equiv K_t(\theta(x), \theta(x'))$.

More generally, we may associate a data point $x$ with a posterior distribution $p(\theta \mid x)$ under a suitable prior. In the case of text, this is one way of "smoothing" the maximum likelihood model, using, for example, a Dirichlet prior. Given a kernel on parameter space, we then average over the posteriors to obtain a kernel on data:

$$K_t(x, x') = \int_M \int_M K_t(\theta, \theta') \, p(\theta \mid x) \, p(\theta' \mid x') \, d\theta \, d\theta' \,. \tag{3}$$

It remains to define the kernel on parameter space. There is a fundamental choice: the kernel associated with heat diffusion on the parameter manifold under the Fisher information metric.

For a manifold $M$ with metric $g_{ij}$ the Laplacian $\Delta : L^2(M) \to L^2(M)$ is given in local coordinates by

$$\Delta = \frac{1}{\sqrt{\det g}} \sum_{ij} \partial_i \sqrt{\det g} \, g^{ij} \, \partial_j \tag{4}$$

where $[g^{ij}] = [g_{ij}]^{-1}$, generalizing the classical operator $\text{div} \circ \nabla = \sum_i \frac{\partial^2}{\partial x_i^2}$. When $M$ is compact the Laplacian has discrete eigenvalues $0 = \lambda_0 < \lambda_1 \leq \lambda_2 \cdots$ with corresponding eigenfunctions $\phi_i$ satisfying $\Delta \phi_i = -\lambda_i \phi_i$. When the manifold has a boundary, appropriate boundary conditions must be imposed in order that $\Delta$ is self-adjoint. Dirichlet boundary conditions set $\phi_i|_{\partial M} = 0$ and Neumann boundary conditions require $\frac{\partial \phi_i}{\partial \nu}\Big|_{\partial M} = 0$ where $\nu$ is the outer normal direction. The following theorem summarizes the basic properties for the kernel of the heat equation $(\Delta - \frac{\partial}{\partial t})u = 0$ on $M$.

**Theorem 1.** *Let $M$ be a geodesically complete Riemannian manifold. Then the heat kernel $K_t(x, y)$ exists and satisfies (1) $K_t(x, y) = K_t(y, x)$, (2) $\lim_{t \to 0} K_t(x, y) = \delta_x(y)$, (3) $\left(\Delta - \frac{\partial}{\partial t}\right) K = 0$, (4) $K_t(x, y) = \int_M K_{t-s}(x, z) K_s(z, y) \, dz$, and (5) $K_t(x, y) = \sum_{i=0}^{\infty} e^{-\lambda_i t} \phi_i(x) \phi_i(y)$.*

We refer to [9] for a proof. Properties 2 and 3 imply that $K_t(x, y)$ solves the heat equation in $x$, starting from $y$. Integrating property 3 against a function $f(y)$ shows that $e^{t\Delta} f(x) = \int_M K_t(x, y) f(y) \, dy$. Therefore, $\int_M \int_M K_t(x, y) f(x) f(y) \, dx \, dy = \int_M f(x) \left(e^{t\Delta} f\right)(x) \, dx = \langle f, e^{t\Delta} f \rangle \geq 0$ since $e^{t\Delta}$ is a positive operator; thus $K_t(x, y)$ is positive definite. Together, these properties show that $K_t$ defines a Mercer kernel. Note that when using such a kernel for classification, the discriminant function $y_t(x) = \sum_i \alpha_i y_i K_t(x, x_i)$ can be interpreted as the solution to the heat equation with initial temperature $y_0(x_i) = \alpha_i y_i$ on labeled data point $x_i$, and $y_0(x) = 0$ on unlabeled points.

The following two basic examples illustrate the geometry of the Fisher information metric and its associated diffusion kernel: the multinomial corresponds to a Riemannian manifold of constant positive curvature, and the spherical normal family to a space of constant negative curvature.

## 2.1 The Multinomial

The multinomial is an important example of how information diffusion kernels can be applied naturally to discrete data. For the multinomial family $\{p(\cdot \,|\, \theta)\}$, $\theta$ is an element of the $d$-simplex, $\sum_{i=1}^{d+1} \theta_i = 1$. The transformation $\theta_i \mapsto 2\sqrt{\theta_i} = z_i$ maps the $d$-simplex to the $d$-sphere of radius 2.

The representation of the Fisher information metric given in equation (2) suggests the geometry underlying the multinomial. In particular, the information metric is given by $g_{ij}(\theta) = \sum_{k=1}^{d+1} \theta_k \partial_i \log \theta_k \partial_j \log \theta_k = \langle \partial_i z, \partial_j z \rangle$ so that the Fisher information corresponds to the inner product of tangent vectors to the sphere, and information geometry for the multinomial is the geometry of the positive orthant of the sphere. The geodesic distance between two points $\theta, \theta'$ is given by

$$d(\theta, \theta') = 2 \arccos \left( \sum_{i=1}^{d+1} \sqrt{\theta_i \theta_i'} \right) . \tag{5}$$

This metric places greater emphasis on points near the boundary, which is expected to be important for text problems, which have sparse statistics. In general for the heat kernel on a Riemannian manifold, there is an asymptotic expansion in terms of the parametrices; see for example [9]. This expands the kernel as

$$K_t(x, y) = (4\pi t)^{-\frac{n}{2}} \exp\left(-\frac{d^2(x, y)}{4t}\right) \sum_{i=0}^{N} \psi_i(x, y) t^i + O(t^N) \tag{6}$$

Using the first order approximation and the explicit distance for the geodesic distance gives

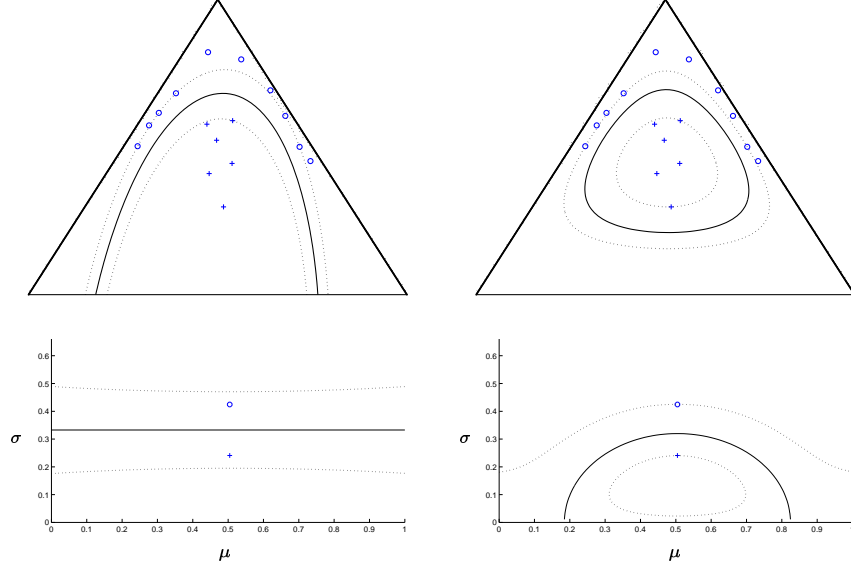

Figure 1: Example decision boundaries using support vector machines with information diffusion kernels for trinomial geometry on the 2-simplex (top right) and spherical normal geometry, $d = 2$ (bottom right), compared with the standard Gaussian kernel (left).

a simple formula for the approximate information diffusion kernel for the multinomial as

$$K_t(\theta, \theta') \approx (4\pi t)^{-\frac{d}{2}} \exp\left(-\frac{1}{t} \arccos^2\left(\sum_{i=1}^{d+1} \sqrt{\theta_i\, \theta_i'}\right)\right) \qquad (7)$$

In Figure 1 this kernel is compared with the standard Euclidean space Gaussian kernel for the case of the trinomial model, $d = 2$.

## 2.2 Spherical Normal

Now consider the statistical family given by $p\left(\cdot \mid \theta = (\mu, \sigma)\right) = \mathcal{N}(\mu, \sigma I_{d-1})$ where $\mu \in \mathbb{R}^{d-1}$ is the mean and $\sigma$ is the scale of the variance. A calculation shows that $g_{ij}(\theta) = \frac{\sqrt{2}}{\sigma^2}\delta_{ij}$. Thus, the Fisher information metric gives $\Theta = \mathbb{R}^{d-1} \times \mathbb{R}_+$ the structure of the upper half plane in hyperbolic space.

The heat kernel on hyperbolic space $\mathbb{H}^d$ has a closed form [2]. For $d = 2m + 1$ it is given by

$$K_t(x, x') = \frac{-1^m}{2^m \pi^m} \frac{1}{\sqrt{4\pi t}} \left(\frac{1}{\sinh \rho} \frac{\partial}{\partial \rho}\right)^m \exp\left(-m^2 t - \frac{\rho^2}{4t}\right) \qquad (8)$$

and for $d = 2m + 2$ the kernel is given by

$$K_t(x, x') = \frac{-1^m}{2^m \pi^m} \frac{\sqrt{2}}{\sqrt{4\pi t}^3} \left(\frac{1}{\sinh \rho} \frac{\partial}{\partial \rho}\right)^m \int_\rho^\infty \frac{s \exp\left(-\frac{(2m+1)^2 t}{4} - \frac{s^2}{4t}\right)}{\sqrt{\cosh s - \cosh \rho}}\, ds \qquad (9)$$

where $\rho = d(x, x')$ is the geodesic distance between the two points in $\mathbb{H}^d$. For $d = 1$ the kernel is identical to the Gaussian kernel on $\mathbb{R}$.

If only the mean $\theta = \mu$ is unspecified, then the associated kernel is the standard Gaussian RBF kernel. In Figure 1 the kernel for hyperbolic space is compared with the Euclidean

space Gaussian kernel for the case of a 1-dimensional normal model with unknown mean and variance, corresponding to $d = 2$. Note that the curved decision boundary for the diffusion kernel makes intuitive sense, since as the variance decreases the mean is known with increasing certainty.

## 3  Spectral Bounds on Covering Numbers

In this section we prove bounds on the entropy and covering numbers for support vector machines that use information diffusion kernels; these bounds in turn yield bounds on the expected risk of the learning algorithms. We adopt the approach of Guo *et al.* [3], and make use of bounds on the spectrum of the Laplacian on a Riemannian manifold, rather than on VC dimension techniques. Our calculations give an indication of how the underlying geometry influences the entropy numbers, which are inverse to the covering numbers.

We begin by recalling the main result of [3], modifying their notation slightly to conform with ours. Let $M \subset \mathbb{R}^d$ be a compact subset of $d$-dimensional Euclidean space, and suppose that $K : M \times M \longrightarrow \mathbb{R}$ is a Mercer kernel. Denote by $\lambda_1 \geq \lambda_2 \geq \cdots \geq 0$ the eigenvalues of $K$, i.e., of the mapping $f \mapsto \int_M K(\cdot, y) f(y) \, dy$, and let $\psi_j(\cdot)$ denote the corresponding eigenfunctions. We assume that $C_K \stackrel{\text{def}}{=} \sup_j \|\psi_j\|_\infty < \infty$.

Given $m$ points $x_i \in M$, the SVM hypothesis class for $\boldsymbol{x} = \{x_i\}$ with weight vector bounded by $R$ is defined as the collection of functions

$$\mathcal{F}_R(\boldsymbol{x}) = \{(x_1, \ldots, x_m) \mapsto (\langle w, \Phi(x_1)\rangle, \ldots \langle w, \Phi(x_m)\rangle), \ \|w\| \leq R\} \qquad (10)$$

where $\Phi(\cdot)$ is the mapping from $M$ to feature space defined by the Mercer kernel, and $\langle \cdot, \cdot \rangle$ and $\|\cdot\|$ denote the corresponding Hilbert space inner product and norm. It is of interest to obtain uniform bounds on the covering numbers $\mathcal{N}(\epsilon, \mathcal{F}_R(\boldsymbol{x}))$, defined as the size of the smallest $\epsilon$-cover of $\mathcal{F}_R(\boldsymbol{x})$ in the metric induced by the norm $\|f\|_{\infty, \boldsymbol{x}} = \max_{i=1,\ldots,m} |f(x_i)|$. The following is the main result of Guo *et al.* [3].

**Theorem 2.**  *Given an integer $n \in \mathbb{N}$, let $j_n^*$ denote the smallest integer for which* $\lambda_{j+1} < \left(\frac{\lambda_1 \cdots \lambda_j}{n^2}\right)^{\frac{1}{j}}$ *and define* $\epsilon_n^* = 6 C_K R \sqrt{j_n^* \left(\frac{\lambda_1 \cdots \lambda_{j_n^*}}{n^2}\right)^{\frac{1}{j_n^*}} + \sum_{i=j_n^*}^\infty \lambda_i}$. *Then* $\sup_{\{x_i\} \in M^m} \mathcal{N}(\epsilon_n^*, \mathcal{F}_R(\boldsymbol{x})) \leq n$.

To apply this result, we will obtain bounds on the indices $j_n^*$ using spectral theory in Riemannian geometry. The following bounds on the eigenvalues of the Laplacian are due to Li and Yau [8].

**Theorem 3.**  *Let $M$ be a compact Riemannian manifold of dimension $d$ with non-negative Ricci curvature, and assume that the boundary of $M$ is convex. Let $0 < \mu_1 \leq \mu_2 \leq \cdots$ denote the eigenvalues of the Laplacian with Dirichlet boundary conditions. Then*

$$c_1(d) \left(\frac{j}{V}\right)^{\frac{2}{d}} \leq \mu_j \leq c_2(d) \left(\frac{j+1}{V}\right)^{\frac{2}{d}} \qquad (11)$$

*where $V$ is the volume of $M$ and $c_1$ and $c_2$ are constants depending only on the dimension.*

Note that the manifold of the multinomial model satisfies the conditions of this theorem. Using these results we can establish the following bounds on covering numbers for information diffusion kernels. We assume Dirichlet boundary conditions; a similar result can be proven for Neumann boundary conditions. We include the constant $V = \text{vol}(M)$ and diffusion coefficient $t$ in order to indicate how the bounds depend on the geometry.

**Theorem 4.** *Let $M$ be a compact Riemannian manifold, with volume $V$, satisfying the conditions of Theorem 3. Then the covering numbers for the Dirichlet heat kernel $\tilde{K}_t$ on $M$ satisfy*

$$\log \mathcal{N}(\epsilon, \mathcal{F}_R(\boldsymbol{x})) = O\left(\left(\frac{V}{t^{\frac{d}{2}}}\right) \log^{\frac{d+2}{2}}\left(\frac{1}{\epsilon}\right)\right) \tag{12}$$

*Proof.* By the lower bound in Theorem 3, the Dirichlet eigenvalues of the heat kernel $K_t(x,y)$, which are given by $\lambda_j = e^{-t\mu_j}$, satisfy $\log \lambda_j \leq -t c_1(d) \left(\frac{j}{V}\right)^{\frac{2}{d}}$. Thus,

$$-\frac{1}{j}\log\left(\frac{\lambda_1 \cdots \lambda_j}{n^2}\right) \geq \frac{t c_1}{j} \sum_{i=1}^{j}\left(\frac{i}{V}\right)^{\frac{2}{d}} + \frac{2}{j}\log n \geq t c_1 \frac{d}{d+2}\left(\frac{j}{V}\right)^{\frac{2}{d}} + \frac{2}{j}\log n \tag{13}$$

where the second inequality comes from $\sum_{i=1}^{j} i^p \geq \int_0^j x^p\, dx = \frac{j^{p+1}}{p+1}$. Now using the upper bound of Theorem 3, the inequality $j_n^* \leq j$ will hold if

$$t c_2 \left(\frac{j+2}{V}\right)^{\frac{2}{d}} \geq -\log \lambda_{j+1} \geq t c_1 \frac{d}{d+2}\left(\frac{j}{V}\right)^{\frac{2}{d}} + \frac{2}{j}\log n \tag{14}$$

or equivalently

$$\frac{t c_2}{V^{\frac{2}{d}}}\left(j(j+2)^{\frac{2}{d}} - \frac{c_1}{c_2}\frac{d}{d+2}j^{\frac{d+2}{d}}\right) \geq 2\log n \tag{15}$$

The above inequality will hold in case

$$j \geq \left\lceil\left(\frac{2V^{\frac{2}{d}}}{t(c_2 - c_1\frac{d}{d+2})}\log n\right)^{\frac{d}{d+2}}\right\rceil \geq \left\lceil\left(\frac{V^{\frac{2}{d}}(d+2)}{t c_1}\log n\right)^{\frac{d}{d+2}}\right\rceil \tag{16}$$

since we may assume that $c_2 \geq c_1$; thus, $j_n^* \leq \left\lceil \bar{c}_1\left(\frac{V^{\frac{2}{d}}}{t}\log n\right)^{\frac{d}{d+2}}\right\rceil$ for a new constant $\bar{c}_1(d)$. Plugging this bound on $j_n^*$ into the expression for $\epsilon_n^*$ in Theorem 2 and using $\sum_{i=j_n^*}^{\infty} e^{-i^{\frac{2}{d}}} = O\left(e^{-j_n^{*\frac{2}{d}}}\right)$, we have after some algebra that $\log\left(\frac{1}{\epsilon_n}\right) = \Omega\left(\left(\frac{t}{V^{\frac{2}{d}}}\right)^{\frac{d}{d+2}}\log^{\frac{2}{d+2}} n\right)$. Inverting the above equation in $\log n$ gives equation (12). $\square$

We note that Theorem 4 of [3] can be used to show that this bound does not, in fact, depend on $m$ and $\boldsymbol{x}$. Thus, for fixed $t$ the covering numbers scale as $\log \mathcal{N}(\epsilon, \mathcal{F}) = O\left(\log^{\frac{d+2}{2}}\left(\frac{1}{\epsilon}\right)\right)$, and for fixed $\epsilon$ they scale as $\log \mathcal{N}(\epsilon, \mathcal{F}) = O\left(t^{-\frac{d}{2}}\right)$ in the diffusion time $t$.

## 4 Experiments

We compared the information diffusion kernel to linear and Gaussian kernels in the context of text classification using the WebKB dataset. The WebKB collection contains some 4000 university web pages that belong to five categories: course, faculty, student, project and staff. A "bag of words" representation was used for all three kernels, using only the word frequencies. For simplicity, all hypertext information was ignored. The information diffusion kernel is based on the multinomial model, which is the correct model under the

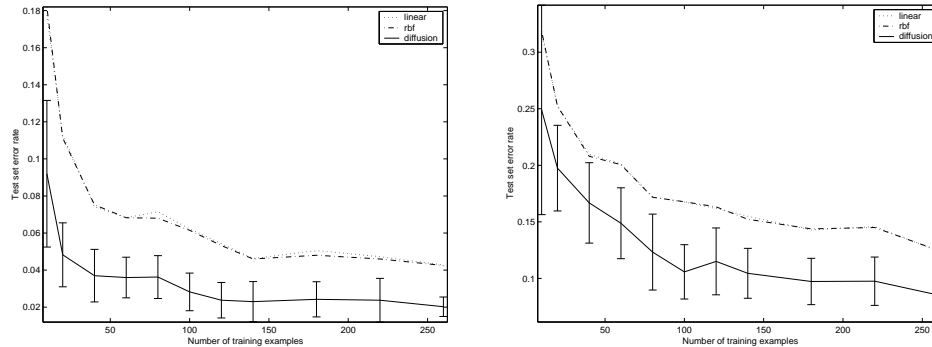

Figure 2: Experimental results on the WebKB corpus, using SVMs for linear (dot-dashed) and Gaussian (dotted) kernels, compared with the information diffusion kernel for the multinomial (solid). Results for two classification tasks are shown, faculty vs. course (left) and faculty vs. student (right). The curves shown are the error rates averaged over 20-fold cross validation.

(incorrect) assumption that the word occurrences are independent. The maximum likelihood mapping $d \mapsto \hat{\theta}(d)$ was used to map a document to a multinomial model, simply normalizing the counts to sum to one.

Figure 2 shows test set error rates obtained using support vector machines for linear, Gaussian, and information diffusion kernels for two binary classification tasks: faculty vs. course and faculty vs. student. The curves shown are the mean error rates over 20-fold cross validation and the error bars represent twice the standard deviation. For the Gaussian and information diffusion kernels we tested values of the kernels' free parameter ($\sigma$ or $\sqrt{t}$) in the set $\{0.1, 0.25, 0.5, 1, 2, 3, 5\}$. The plots in Figure 2 use the best parameter value in the above range.

Our results are consistent with previous experiments on this dataset [5], which have observed that the linear and Gaussian kernels result in very similar performance. However the information diffusion kernel significantly outperforms both of them, almost always obtaining lower error rate than the average error rate of the other kernels. For the faculty vs. course task, the error rate is halved. This result is striking because the kernels use identical representations of the documents, vectors of word counts (in contrast to, for example, string kernels). We attribute this improvement to the fact that the information metric places more emphasis on points near the boundary of the simplex.

## 5  Discussion

Kernel-based methods generally are "model free," and do not make distributional assumptions about the data that the learning algorithm is applied to. Yet statistical models offer many advantages, and thus it is attractive to explore methods that combine data models and purely discriminative methods for classification and regression. Our approach brings a new perspective to combining parametric statistical modeling with non-parametric discriminative learning. In this aspect it is related to the methods proposed by Jaakkola and Haussler [4]. However, the kernels we investigate here differ significantly from the Fisher kernel proposed in [4]. In particular, the latter is based on the Fisher score $\nabla_{\theta} \log p(X \mid \hat{\theta})$ at a single point $\hat{\theta}$ in parameter space, and in the case of an exponential family model it is given by a covariance $K_F(x, x') = \sum_i (x_i - E_{\hat{\theta}}[X_i]) (x'_i - E_{\hat{\theta}}[X_i])$. In contrast, infor-

mation diffusion kernels are based on the full geometry of the statistical family, and yet are also invariant under reparameterization of the family.

Bounds on the covering numbers for information diffusion kernels were derived for the case of positive curvature, which apply to the special case of the multinomial. We note that the resulting bounds are essentially the same as those that would be obtained for the Gaussian kernel on the flat $d$-dimensional torus, which is the standard way of "compactifying" Euclidean space to get a Laplacian having only discrete spectrum; the results of [3] are formulated for the case $d = 1$, corresponding to the circle $S^1$. Similar bounds for general manifolds with curvature bounded below by a negative constant should also be attainable.

While information diffusion kernels are very general, they may be difficult to compute in particular cases; explicit formulas such as equations (8–9) for hyperbolic space are rare. To approximate an information diffusion kernel it may be attractive to use the parametrices and geodesic distance $d(\theta, \theta')$ between points, as we have done for the multinomial. In cases where the distance itself is difficult to compute exactly, a compromise may be to approximate the distance between nearby points in terms of the Kullback-Leibler divergence, using the relation $d^2(\theta, \theta') \approx D(p(\cdot \mid \theta) \,\|\, p(\cdot \mid \theta'))$.

The primary "degree of freedom" in the use of information diffusion kernels lies in the specification of the mapping of data to model parameters, $x \mapsto \theta(x)$. For the multinomial, we have used the maximum likelihood mapping $x \mapsto \hat{\theta}(x) = \arg\max_\theta p(x \mid \theta)$, which is simple and well motivated. As indicated in Section 2, there are other possibilities. This remains an interesting area to explore, particularly for latent variable models.

## Acknowledgements

This work was supported in part by NSF grant CCR-0122581.

## References

[1] S. Amari and H. Nagaoka. *Methods of Information Geometry*, volume 191 of *Translations of Mathematical Monographs*. American Mathematical Society, 2000.

[2] A. Grigor'yan and M. Noguchi. The heat kernel on hyperbolic space. *Bulletin of the London Mathematical Society*, 30:643–650, 1998.

[3] Y. Guo, P. L. Bartlett, J. Shawe-Taylor, and R. C. Williamson. Covering numbers for support vector machines. *IEEE Trans. Information Theory*, 48(1), January 2002.

[4] T. S. Jaakkola and D. Haussler. Exploiting generative models in discriminative classifiers. In *Advances in Neural Information Processing Systems*, volume 11, 1998.

[5] T. Joachims, N. Cristianini, and J. Shawe-Taylor. Composite kernels for hypertext categorisation. In *Proceedings of the International Conference on Machine Learning (ICML)*, 2001.

[6] R. E. Kass and P. W. Vos. *Geometrical Foundations of Asymptotic Inference*. Wiley Series in Probability and Statistics. John Wiley & Sons, 1997.

[7] R. I. Kondor and J. Lafferty. Diffusion kernels on graphs and other discrete input spaces. In *Proceedings of the International Conference on Machine Learning (ICML)*, 2002.

[8] P. Li and S.-T. Yau. Estimates of eigenvalues of a compact Riemannian manifold. In *Geometry of the Laplace Operator*, volume 36 of *Proceedings of Symposia in Pure Mathematics*, pages 205–239, 1980.

[9] R. Schoen and S.-T. Yau. *Lectures on Differential Geometry*, volume 1 of *Conference Proceedings and Lecture Notes in Geometry and Topology*. International Press, 1994.
